# Multiclass Boosting: Theory and Algorithms

**Mohammad J. Saberian**
Statistical Visual Computing Laboratory,
University of California, San Diego
saberian@ucsd.edu

**Nuno Vasconcelos**
Statistical Visual Computing Laboratory,
University of California, San Diego
nuno@ucsd.edu

## Abstract

The problem of multi-class boosting is considered. A new framework, based on multi-dimensional codewords and predictors is introduced. The optimal set of codewords is derived, and a margin enforcing loss proposed. The resulting risk is minimized by gradient descent on a multidimensional functional space. Two algorithms are proposed: 1) CD-MCBoost, based on coordinate descent, updates one predictor component at a time, 2) GD-MCBoost, based on gradient descent, updates all components jointly. The algorithms differ in the weak learners that they support but are both shown to be 1) Bayes consistent, 2) margin enforcing, and 3) convergent to the global minimum of the risk. They also reduce to AdaBoost when there are only two classes. Experiments show that both methods outperform previous multiclass boosting approaches on a number of datasets.

## 1 Introduction

Boosting is a popular approach to classifier design in machine learning. It is a simple and effective procedure to combine many weak learners into a strong classifier. However, most existing boosting methods were designed primarily for binary classification. In many cases, the extension to $M$-ary problems (of $M > 2$) is not straightforward. Nevertheless, the design of multi-class boosting algorithms has been investigated since the introduction of AdaBoost in [8].

Two main approaches have been attempted. The first is to reduce the multiclass problem to a collection of binary sub-problems. Methods in this class include the popular "one vs all" approach, or methods such as "all pairs", ECOC [4, 1], AdaBoost-M2 [7], AdaBoost-MR [18] and AdaBoost-MH [18, 9]. The binary reduction can have various problems, including 1) increased complexity, 2) lack of guarantees of an optimal joint predictor, 3) reliance on data representations, such as adding one extra dimension that includes class numbers to each data point [18, 9], that may not necessarily enable effective binary discrimination, or 4) using binary boosting scores that do not represent true class probabilities [15]. The second approach is to boost an $M$-ary classifier directly, using multiclass weak learners, such as trees. Methods of this type include AdaBoost-M1[7], SAMME[12] and AdaBoost-Cost [16]. These methods require strong weak learners which substantially increase complexity and have high potential for overfitting. This is particularly problematic because, although there is a unified view of these methods under the game theory interpretation of boosting [16], none of them has been shown to maximize the multiclass margin. Overall, the problem of optimal and efficient $M$-ary boosting is still not as well understood as its binary counterpart.

In this work, we introduce a new formulation of multi-class boosting, based on 1) an alternative definition of the margin for $M$-ary problems, 2) a new loss function, 3) an optimal set of codewords, and 4) the statistical view of boosting, which leads to a convex optimization problem in a multidimensional functional space. We propose two algorithms to solve this optimization: CD-MCBoost, which is a functional coordinate descent procedure, and GD-MCBoost, which implements functional gradient descent. The two algorithms differ in terms of the strategy used to update the multidimensional predictor. CD-MCBoost supports any type of weak learners, updating one component of the predictor per boosting iteration, GD-MCBoost requires multiclass weak learners but updates all

components simultaneously. Both methods directly optimize the predictor of the multiclass problem and are shown to be 1) Bayes consistent, 2) margin enforcing, and 3) convergent to the global minimum of the classification risk. They also reduce to AdaBoost for binary problems. Experiments show that they outperform comparable prior methods on a number of datasets.

## 2 Multiclass boosting

We start by reviewing the fundamental ideas behind the classical use of boosting for the design of *binary* classifiers, and then extend these ideas to the multiclass setting.

### 2.1 Binary classification

A binary classifier, $F(x)$, is a mapping from examples $x \in \mathcal{X}$ to class labels $y \in \{-1, 1\}$. The optimal classifier, in the minimum probability of error sense, is Bayes decision rule

$$F(x) = \arg\min_{y \in \{-1,1\}} P_{Y|X}(y|x). \tag{1}$$

This can be hard to implement, due to the difficulty of estimating the probabilities $P_{Y|X}(y|x)$. This difficulty is avoided by large margin methods, such as boosting, which implement the classifier as

$$F(x) = sign[f^*(x)] \tag{2}$$

where $f^*(x) : \mathcal{X} \to \mathbb{R}$ is the continuous valued predictor

$$f^*(x) = \arg\min_f R(f) \tag{3}$$

that minimizes the classification risk

$$R(f) = E_{X,Y}\{L[y, f(x)]\} \tag{4}$$

associated with a loss function $L[.,.]$. In practice, the optimal predictor is learned from a sample $\mathcal{D} = \{(x_i, y_i)\}_{i=1}^n$ of training examples, and (4) is approximated by the empirical risk

$$R(f) \approx \sum_{i=1}^n L[y_i, f(x_i)]. \tag{5}$$

The loss $L[.,.]$ is said to be Bayes consistent if (1) and (2) are equivalent. For large margin methods, such as boosting, the loss is also a function of the classification margin $yf(x)$, i.e.

$$L[y, f(x)] = \phi(yf(x)) \tag{6}$$

for some non-negative function $\phi(.)$. This dependence on the margin $yf(x)$ guarantees that the classifier has good generalization when the training sample is small [19]. Boosting learns the optimal predictor $f^*(x) : \mathcal{X} \to \mathbb{R}$ as the solution of

$$\begin{cases} \min_{f(x)} & R(f) \\ s.t & f(x) \in span(\mathcal{H}). \end{cases} \tag{7}$$

where $\mathcal{H} = \{h_1(x), ...h_p(x)\}$ is a set of weak learners $h_i(x) : \mathcal{X} \to \mathbb{R}$, and the optimization is carried out by gradient descent in the functional space $span(\mathcal{H})$ of linear combinations of $h_i(x)$ [14].

### 2.2 Multiclass setting

To extend the above formulation to the multiclass setting, we note that the definition of the classification labels as $\pm 1$ plays a significant role in the formulation of the binary case. One of the difficulties of the multiclass extension is that these labels do not have an immediate extension to the multiclass setting. To address this problem, we return to the classical setting, where the class labels of a $M$-ary problem take values in the set $\{1, \ldots, M\}$. Each class $k$ is then mapped into a distinct class label $y^k$, which can be thought of as a *codeword* that identifies the class.

In the binary case, these codewords are defined as $y^1 = 1$ and $y^2 = -1$. It is possible to derive an alternative form for the expressions of the margin and classifier $F(x)$ that depends explicitly on codewords. For this, we note that (2) can be written as

$$F(x) = \arg\max_k y^k f^*(x) \tag{8}$$

and the margin can be expressed as

$$yf = \begin{cases} f & \text{if } k = 1 \\ -f & \text{if } k = 2 \end{cases} = \begin{cases} \frac{1}{2}(y^1 f - y^2 f) & \text{if } k = 1 \\ \frac{1}{2}(y^2 f - y^1 f) & \text{if } k = 2 \end{cases} = \frac{1}{2}(y^k f - \max_{l \neq k} y^l f). \tag{9}$$

The interesting property of these forms is that they are directly extensible to the $M$-ary classification case. For this, we assume that the codewords $y^k$ and the predictor $f(x)$ are multi-dimensional, i.e. $y^k, f(x) \in \mathbb{R}^d$ for some dimension $d$ which we will discuss in greater detail in the following section. The margin of $f(x)$ with respect to class $k$ is then defined as

$$\mathcal{M}(f(x), y^k) = \frac{1}{2}[< f(x), y^k > - \max_{l \neq k} < f(x), y^l >] \tag{10}$$

and the classifier as

$$F(x) = \arg\max_k < f(x), y^k >, \tag{11}$$

where $< ., . >$ is the standard dot-product. Note that this is equivalent to

$$F(x) = \arg\max_{k \in \{1,...,M\}} \mathcal{M}(f(x), y^k), \tag{12}$$

and thus $F(x)$ is the class of largest margin for the predictor $f(x)$. This definition is closely related to previous notions of multiclass margin. For example, it generalizes that of [11], where the codewords $y^k$ are restricted to the binary vectors in the canonical basis of $\mathbb{R}^d$, and is a special case of that in [1], where the dot products $< f(x), y^k >$ are replaced by a generic function of $f, x$, and $k$. Given a training sample $\mathcal{D} = \{(x_i, y_i)\}_{i=1}^n$, the optimal predictor $f^*(x)$ minimizes the risk

$$R_M(f) = E_{X,Y}\{L_M[y, f(x)]\} \approx \sum_{i=1}^n L_M[y_i, f(x_i)]\} \tag{13}$$

where $L_M[., .]$ is a multiclass loss function. A natural extension of (6) and (9) is a loss of the form

$$L_M[y, f(x)] = \phi(\mathcal{M}(f(x), y)). \tag{14}$$

To avoid the nonlinearity of the max operator in (10), we rely on

$$L_M[y, f(x)] = \sum_{k=1}^M e^{-\frac{1}{2}[<f(x),y> - <f(x),y^k>]}. \tag{15}$$

which is shown, in Appendix A, to upper bound $1 + e^{-\mathcal{M}(f(x),y)}$. It follows that the minimization of the risk of (13) encourages predictors of large margin $\mathcal{M}(f^*(x_i), y_i), \forall i$. For $M = 2$, $L_M[y, f(x)]$ reduces to

$$L_2[y, f(x)] = 1 + e^{-yf(x)} \tag{16}$$

and the risk minimization problem is identical to that of AdaBoost [8]. In appendices B and C it is shown that $R_M(f)$ is convex and Bayes consistent, in the sense that if $f^*(x)$ is the minimizer of (13), then

$$< f^*(x), y^k >= \log P_{Y|X}(y^k|x) + c \quad \forall k \tag{17}$$

and (11) implements the Bayes decision rule

$$F(x) = \arg\max_k P_{Y|X}(y^k|x). \tag{18}$$

## 2.3 Optimal set of codewords

From (15), the choice of codewords $y^k$ has an impact in the optimal predictor $f^*(x)$, which is determined by the projections $< f^*(x), y^k >$. To maximize the margins of (10), the difference between these projections should be as large as possible. To accomplish this we search for the set of $M$ distinct unit codewords $\mathcal{Y} = \{y^1, \ldots, y^M\} \in \mathbb{R}^d$ that are as dissimilar as possible

$$\begin{cases} \max_{d, y^1, \ldots y^M} [\min_{i \neq j} ||y^i - y^j||^2] \\ \\ s.t \quad ||y^k|| = 1 \quad \forall k = 1..M. \\ \quad\quad y^k \in \mathbb{R}^d \quad \forall k = 1..M. \end{cases} \tag{19}$$

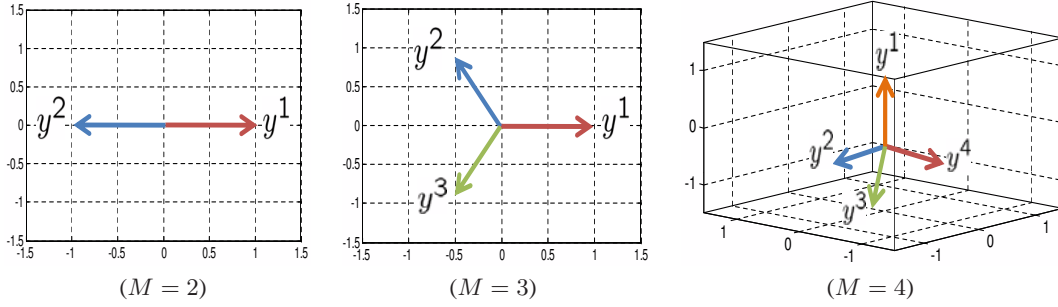

$$(M = 2) \qquad (M = 3) \qquad (M = 4)$$

Figure 1: Optimal codewords for $M = 2, 3, 4$.

To solve this problem, we start by noting that, for $d < M$, the smallest distance of (19) can be increased by simply increasing $d$, since this leads to a larger space. On the other hand, since $M$ points $y^1, ... y^M$ lie in an, at most, $M - 1$ dimensional subspace of $\mathbb{R}^d$, e.g. any three points belong to a plane, there is no benefit in increasing $d$ beyond $M - 1$. On the contrary, as shown in Appendix D, if $d > M - 1$ there exits a vector $v \in \mathbb{R}^d$ with equal projection on all codewords,

$$< y^i, v >=< y^j, v > \qquad \forall i, j = 1, .., M. \qquad (20)$$

Since the addition of $v$ to the predictor $f(x)$ does not change the classification rule of (11), this makes the optimal predictor underdetermined. To avoid this problem, we set $d = M - 1$. In this case, as shown in Appendix E, the vertices of a $M - 1$ dimensional *regular simplex*[1] centered at the origin [3] are solutions of (19). Figure 1 presents the set of optimal codewords when $M = 2, 3, 4$. Note that in the binary case this set consists of the traditional codewords $y_i \in \{+1, -1\}$. In general, there is no closed form solution for the vertices of a regular simplex of $M$ vectors. However, these can be derived from those of a regular simplex of $M - 1$ vectors, and a recursive solution is possible [3].

# 3   Risk minimization

We have so far defined a proper margin loss function for $M$-ary classification and identified an optimal codebook. In this section, we derive two boosting algorithms for the minimization of the classification risk of (13). These algorithms are both based on the GradientBoost framework [14]. The first is a functional coordinate descent algorithm, which updates a single component of the predictor per boosting iteration. The second is a functional gradient descent algorithm that updates all components simultaneously.

## 3.1   Coordinate descent

In the first method, each component $f_i^*(x)$ of the optimal predictor $f^*(x) = [f_1^*(x), ..f_{M-1}^*(x)]$, is the linear combination of weak learners that solves the optimization problem

$$\begin{cases} \min_{f_1(x), ..., f_{M-1}(x)} & R([f_1(x), ..., f_{M-1}(x)]) \\ s.t & f_j(x) \in span(\mathcal{H}) \quad \forall j = 1..M - 1. \end{cases} \qquad (21)$$

where $\mathcal{H} = \{h_1(x), ... h_p(x)\}$ is a set of weak learners, $h_i(x) : \mathcal{X} \rightarrow \mathbb{R}$. These can be stumps, regression trees, or member of any other suitable model family. We denote by $f^t(x) = [f_1^t(x), ..., f_{M-1}^t(x)]$ the predictor available after $t$ boosting iterations. At iteration $t + 1$ a single component $f_j(x)$ of $f(x)$ is updated with a step in the direction of the scalar functional $g$ that most decreases the risk $R[f_1^t, ..., f_j^t + \alpha_j^* g, ..., f_{M-1}^t]$. For this, we consider the functional derivative of $R[f(x)]$ along the direction of the functional $g : \mathcal{X} \rightarrow \mathbb{R}$, at point $f(x) = f^t(x)$, with respect to the $j^{th}$ component $f_j(x)$ of $f(x)$ [10],

$$\delta R[f^t; j, g] = \left. \frac{\partial R[f^t + \epsilon g \mathbf{1}_j]}{\partial \epsilon} \right|_{\epsilon=0}, \qquad (22)$$

where $\mathbf{1}_j \in \mathbb{R}^d$ is a vector whose $j^{th}$ element is one and the remainder zero, i.e. $f^t + \epsilon g \mathbf{1}_j = [f_1^t, .., f_j^t + \epsilon g, .. f_{M-1}^t]$. Using the risk of (13), it is shown in Appendix F that

$$-\delta R[f^t; j, g] \quad = \quad \sum_{i=1}^n g(x_i) w_i^j, \tag{23}$$

with

$$w_i^j = \frac{1}{2} e^{-\frac{1}{2} <f^t(x_i), y_i>} \sum_{k=1}^M < \mathbf{1}_j, y_i - y^k > e^{\frac{1}{2} <f^t(x_i), y^k>}. \tag{24}$$

The direction of greatest risk decrease is the weak learner

$$g_j^*(x) = \arg\max_{g \in \mathcal{H}} \sum_{i=1}^n g(x_i) w_i^j, \tag{25}$$

and the optimal step size along this direction

$$\alpha_j^* = \arg\min_{\alpha \in \mathbb{R}} R[f^t(x) + \alpha g_j^*(x) \mathbf{1}_j]. \tag{26}$$

The classifier is thus updated as

$$f^{t+1} = f^t(x) + \alpha_j^* g_j^*(x) \mathbf{1}_j = [f_1^t, ..., f_j^t + \alpha_j^* g_j^*, ..., f_{M-1}^t] \tag{27}$$

This procedure is summarized in Algorithm 1-left and denoted CD-MCBoost. It starts with $f^0(x) = 0 \in \mathbb{R}^d$ and updates the predictor components sequentially. Note that, since (13) is a convex function of $f(x)$, it converges to the global minimum of the risk.

## 3.2 Gradient descent

Alternatively, (13) can be minimized by learning a linear combination of multiclass weak learners. In this case, the optimization problem is

$$\begin{cases} \min_{f(x)} & R[f(x)] \\ s.t & f(x) \in span(\overline{\mathcal{H}}), \end{cases} \tag{28}$$

where $\overline{\mathcal{H}} = \{\overline{h}_1(x), ..., \overline{h}_p(x)\}$ is a set of multiclass weak learners, $\overline{h}_i(x) : \mathcal{X} \to \mathbb{R}^{M-1}$, such as decision trees. Note that to fit tree classifiers in this definition their output (usually a class number) should be translated into a class codeword. As before, let $f^t(x) \in \mathbb{R}^{M-1}$ be the predictor available after $t$ boosting iterations. At iteration $t + 1$ a step is given along the direction $g(x) \in \overline{\mathcal{H}}$ of largest decrease of the risk $R[f(x)]$. For this, we consider the directional functional derivative of $R[f(x)]$ along the direction of the functional $g : \mathcal{X} \to \mathbb{R}^{M-1}$, at point $f(x) = f^t(x)$.

$$\delta R[f^t; g] \quad = \quad \frac{\partial R[f^t + \epsilon g]}{\partial \epsilon}\bigg|_{\epsilon=0}. \tag{29}$$

As shown in Appendix G,

$$-\delta R[f^t; g] \quad = \quad \sum_{i=1}^n < g(x_i), w_i > \tag{30}$$

where $w_i \in \mathbb{R}^{M-1}$

$$w_i \quad = \quad \frac{1}{2} e^{-\frac{1}{2} <f^t(x_i), y_i>} \sum_{k=1}^M (y_i - y^k) e^{\frac{1}{2} <f^t(x_i), y^k>}. \tag{31}$$

The direction of greatest risk decrease is the weak learner

$$g^*(x) = \arg\max_{g \in \overline{\mathcal{H}}} \sum_{i=1}^n < g(x_i), w_i >, \tag{32}$$

and the optimal step size along this direction

$$\alpha^* = \arg\min_{\alpha \in \mathbb{R}} R[f^t(x) + \alpha g^*(x)]. \tag{33}$$

The predictor is updated to $f^{t+1}(x) = f^t(x) + \alpha^* g^*(x)$. This procedure is summarised in Algorithm 1-right, and denoted GD-MCBoost. Since (13) is convex, it converges to the global minimum of the risk.

**Algorithm 1 CD-MCBoost and GD-MCBoost**

---

**Input:** Number of classes $M$, set of codewords $\mathcal{Y} = \{y^1, \ldots, y^M\}$, number of iterations $N$ and dataset $S = \{(x_1, y_1), ..., (x_n, y_n)\}$, where $x_i$ are examples and $y_i \in \mathcal{Y}$ are their class codewords.
**Initialization:** set $t = 0$, and $f^t = 0 \in \mathbb{R}^{M-1}$

*CD-MCBoost*                                        *GD-MCBoost*

   **while** $t < N$ **do**
     **for** $j = 1$ to $M - 1$ **do**
       Compute $w_i^j$ with (24)                 **while** $t < N$ **do**
       Find $g_j^*(x)$, $\alpha_j^*$ using (25) and (26)        Compute $w_i$ with (31)
       Update $f_j^{t+1}(x) = f_j^t(x) + \alpha_j^* g_j^*(x)$     Find $g^*(x)$, $\alpha^*$ using (32) and (33)
       Update $f_k^{t+1}(x) = f_k^t(x)$   $\forall k \neq j$     Update $f^{t+1}(x) = f^t(x) + \alpha^* g^*(x)$
       $t = t + 1$                            $t = t + 1$
     **end for**                               **end while**
   **end while**

 **Output:** decision rule: $F(x) = \arg\max_k < f^N(x), y^k >$

---

## 4   Comparison to previous methods

Multi-dimensional predictors and codewords have been used implicitly, [7, 18, 16, 6], or explicitly, [12, 9], in all previous multiclass boosting methods.

**"one vs all", "all pairs" and "ECOC" [1]:** as shown in [1], these methods can be interpreted as assigning a codeword $y^k$ to each class, where $y^k \in \{+1, 0, -1\}^l$ and $l = M$ for "one vs all", $l = \frac{M(M-1)}{2}$ for "all pairs" and $l$ is variable for "ECOC", depending on the error correction code. In all these methods, binary classifiers are learned independently for each of the codeword components. This does not guarantee an optimal joint predictor. These methods are similar to CD-MCBoost in the sense that the predictor components are updated individually at each boosting iteration. However, in CD-MCBoost, the codewords are not restricted to $\{+1, 0, -1\}$ and the predictor components are learned jointly.

**AdaBoost-MH [18, 9]:** This method converts the $M$-ary classification problem into a binary one, learned from a $M$ times larger training set, where each example $x$ is augmented with a feature $y$ that identifies a class. Examples such that $x$ belongs to class $y$ receive binary label 1, while the remaining receive the label $-1$ [9]. In this way, the binary classifier learns if the multiclass label $y$ is correct for $x$ or not. AdaBoost-MH uses weak learners $h_t : \mathcal{X} \times \{1, \ldots, M\} \to \mathbb{R}$ and the decision rule

$$\bar{F}(x) = \arg\max_{j \in \{1, 2, ..M\}} \sum_t h_t(x, j) \tag{34}$$

where $t$ is the iteration number. This is equivalent to the decision rule of (11) if $f(x)$ is an $M$-dimensional predictor with $j^{th}$ component $f_j(x) = \sum_t h_t(x, j)$, and the label codewords are defined as $y^j = \mathbf{1}_j$. This method is comparable to CD-MCBoost in the sense that it does not require multiclass weak learners. However, there are no guarantees that the weak learners in common use are able to discriminate the complex classes of the augmented binary problem.

**AdaBoost-M1 [7] and AdaBoost-Cost [16]:** These methods use multiclass weak learners $h_t : \mathcal{X} \to \{1, 2, ..M\}$ and a classification rule of the form

$$\bar{F}(x) = \arg\max_{j \in \{1, 2, ..M\}} \sum_{t | h_t(x) = j} \alpha_t h_t(x), \tag{35}$$

where $t$ is the boosting iteration and $\alpha_t$ the coefficient of weak learner $h_t(x)$. This is equivalent to the decision rule of (11) if $f(x)$ is an $M$-dimensional predictor with $j^{th}$ component $f_j(x) = \sum_{t | h_t(x) = j} \alpha_t h_t(x)$ and label codewords $y^j = \mathbf{1}_j$. These methods are comparable to GD-MCBoost, in the sense that they update the predictor components simultaneously. However, they have not been shown to be Bayes consistent, and it is not clear that they can be interpreted as maximizing the multiclass margin.

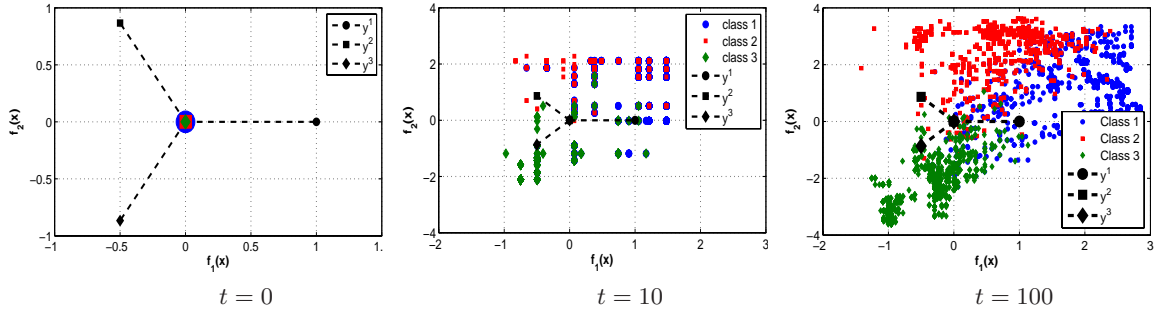

$$t = 0 \qquad\qquad t = 10 \qquad\qquad t = 100$$

Figure 2: Classifier predictions of CD-MCBoost, on the test set, after $t = 0, 10, 100$ boosting iterations.

**SAMME [12]:** This method explicitly uses $M$-dimensional predictors with codewords

$$y^j = \frac{M\mathbf{1}_j - \mathbf{1}}{M - 1} = \left[ \frac{-1}{M-1}, \frac{-1}{M-1}, ..., 1, \frac{-1}{M-1}, \frac{-1}{M-1} \right] \in \mathbb{R}^M, \qquad (36)$$

and decision rule

$$\bar{F}(x) = \arg \max_{j \in \{1,2,..M\}} f_j(x). \qquad (37)$$

Since, as discussed in Section 2.3, the optimal detector is not unique when the predictor is $M$-dimensional, this algorithm includes the additional constraint $\sum_{j=1}^{M} f_j(x) = 0$ and solves a constrained optimization problem [12, 9]. It is comparable to GD-MCBoost in the sense that it updates the predictor components simultaneously, but uses the loss function $L_{SAMME}[y^k, f(x)] = e^{-\frac{1}{M} <y^k, f(x)>}$. Using (36), the minimization of this loss is equivalent to maximizing

$$\mathcal{M}'(f(x), y^k) = <f(x), y^k> = f_k(x) - \frac{1}{M-1} \sum_{j \neq k} f_j(x), \qquad (38)$$

which is not a proper margin since $\mathcal{M}'(f(x), y^k) > 0$ does not imply correct classification i.e. $f_k(x) > f_j(x) \quad \forall j \neq k$. Hence, SAMME does not guarantee a large margin solution for the multiclass problem.

When compared to all these methods, MCBoost has the advantage of combining 1) a Bayes consistent and margin enforcing loss function, 2) an optimal set of codewords, 3) the ability to boost any type of weak learner, 4) guaranteed convergence to the global minimum of (21), for CD-MCBoost, or (28), for GD-MCBoost, and 5) equivalence to the classical AdaBoost algorithm for binary problems. It is worth emphasizing that *MCBoost can boost any type of weak learners of non-zero directional derivative*, i.e. non-zero (23) for CD-MCBoost and (30) for GD-MCBoost. This is independent of the type of weak learner output, and unlike previous multiclass boosting approaches, which can only boost weak learners of specific output types. Note that, although the weak learner selection criteria of previous approaches can have interesting interpretations, e.g. based on weighted error rates [16], these only hold for specific weak learners. Finally, MCBoost extends the definition of margin and loss function to multi-dimensional predictors. The derivation of Section 2 can easily be generalized to the design of other multiclass boosting algorithms by the use of 1) alternative $\phi(v)$ functions in (14) (e.g. those of the logistic [9] or Tangent [13] losses for increased outlier robustness, asymmetric losses for cost-sensitive classification, etc.), and 2) alternative optimization approaches (e.g. Newton's method [9, 17]).

## 5 Evaluation

A number of experiments were conducted to evaluate the MCBoost algorithms[2].

### 5.1 Synthetic data

We start with a synthetic example, for which the optimal decision rule is known. This is a three class problem, with two-dimensional Gaussian classes of means $[1, 2], [-1, 0], [2, -1]$ and covariances of

Table 1: Accuracy of multiclass boosting methods, using decision stumps, on six UCI data sets

| method | landsat | letter | pendigit | optdigit | shuttle | isolet |
|---|---|---|---|---|---|---|
| *One Vs All* | 84.80% | **50.92%** | 86.56% | 89.93% | 87.11% | 88.97% |
| *AdaBoost-MH* [18] | 47.70% | 15.73% | 24.41% | 73.62% | 79.16% | 66.71% |
| *CD-MCBoost* | **85.70%** | 49.60% | **89.51%** | **92.82%** | **88.01%** | **91.02%** |

Table 2: Accuracy of multiclass boosting methods, using trees of max depth 2, on six UCI data sets

| method | landsat | letter | pendigit | optdigit | shuttle | isolet |
|---|---|---|---|---|---|---|
| *AdaBoost-M1*[7] | 72.85% | – | – | – | 96.45% | – |
| *AdaBoost-SAMME*[12] | 79.80% | 45.65% | 83.82% | 87.53% | 99.70% | 61.00% |
| *AdaBoost-Cost* [16] | 83.95% | 42.00% | 80.53% | 86.20% | 99.55% | 63.69% |
| *GD-MCBoost* | **86.65%** | **59.65%** | **92.94%** | **92.32%** | **99.73%** | **84.28%** |

$[1, 0.5; 0.5, 2], [1, 0.3; 0.3, 1], [.4, 0.1; 0.1, 0.8]$ respectively. Training and test sets of $1,000$ examples each were randomly sampled and the Bayes rule computed in closed form [5]. The associated Bayes error rate was $11.67\%$ in the training and $11.13\%$ in the test set. A classifier was learned with CD-MCBoost and decision stumps.

Figure 2) shows predictions[3] of $f^t(x)$ on the test set, for $t = 0, 10, 100$. Note that $f^0(x_i) = [0, 0]$ for all examples $x_i$. However, as the iterations proceed, CD-MCBoost produces predictions that are more aligned with the true class codewords, shown as dashed lines, while maximizing the distance between examples of different classes (by increasing their distance to the origin). In this context, "alignment of $f(x)$ with $y^k$" implies that $< f(x), y^k > \geq < f(x), y^j >, \forall j \neq k$. This combination of alignment and distance maximization results in higher margins, leading to more accurate and robust classification. The test error rate after 100 iterations of boosting was $11.30\%$, and very close to the Bayes error rate of $11.13\%$.

## 5.2 CD-MCBoost

We next conducted a number of experiments to evaluate the performance of CD-MCBoost on the six UCI datasets of Table 1. Among the methods identified as comparable in the previous section, we implemented "one vs all" and AdaBoost-MH [18]. In all cases, decision stumps were used as weak learners, and we used the training/test set decomposition specified for each dataset. The "one vs all" detectors were trained with 20 iterations. The remaining methods were then allowed to include the same number of weak learners in their final decision rules. Table 1 presents the resulting classification accuracies. CD-MCBoost produced the most accurate classifier in four of the five datasets, and was a close second in the remaining one. "One vs all" achieved the next best performance, with AdaBoost-MH producing the worst classifiers.

## 5.3 GD-MCBoost

Finally, the performance of GD-MCBoost was compared to AdaBoost-M1 [7], AdaBoost-Cost [16] and AdaBoost-SAMME [12]. The experiments were based on the UCI datasets of the previous section, but the weak learners were now trees of depth 2. These were built with a greedy procedure so as to 1) minimize the weighted error rate of AdaBoost-M1 [7] and AdaBoost-SAMME[12], 2) minimize the classification cost of AdaBoost-Cost [16], or 3) maximize (32) for GD-MCBoost. Table 2 presents the classification accuracy of each method, for 50 training iterations. GD-MCBoost achieved the best accuracy on all datasets, reaching substantially larger classification rate than all other methods in the most difficult datasets, e.g. from a previous best of $63.69\%$ to $84.28\%$ in isolet, $45.65\%$ to $59.65\%$ in letter, and $83.82\%$ to $92.94\%$ in pendigit. Among the remaining methods, AdaBoost-SAMME achieved the next best performance, although this was close to that of AdaBoost-Cost. AdaBoost-M1 had the worst results, and was not able to boost the weak learners used in this experiment for four of the six datasets. It should be noted that the results of Tables 1 and 2 are not directly comparable, since the classifiers are based on different types of weak learners and have different complexities.

## Footnotes

[1]A regular $M - 1$ dimensional simplex is the convex hull of $M$ normal vectors which have equal pair-wise distances.

[2]Codes for CD-MCBoost and GD-MCBoost are available from [2].

[3]We emphasize the fact that these are plots of $f^t(x) \in \mathbb{R}^2$, not $x \in \mathbb{R}^2$.

# References

[1] E. L. Allwein, R. E. Schapire, and Y. Singer. Reducing multiclass to binary: a unifying approach for margin classifiers. *J. Mach. Learn. Res.*, 1:113–141, September 2001.

[2] N. N. Author. Suppressed for anonymity.

[3] H. S. M. Coxeter. *Regular Polytopes*. Dover Publications, 1973.

[4] T. G. Dietterich and G. Bakiri. Solving multiclass learning problems via error-correcting output codes. *Journal of Artificial Intelligence Research*, 2:263–286, 1995.

[5] R. O. Duda, P. E. Hart, and D. G. Stork. *Pattern Classification*. Wiley, New York, 2. edition, 2001.

[6] G. Eibl and R. Schapire. Multiclass boosting for weak classifiers. In *Journal of Machine Learning Research*, pages 6–189, 2005.

[7] Y. Freund and R. E. Schapire. Experiments with a new boosting algorithm. In *Proceedings of the Thirteenth International Conference In Machine Learning*, pages 148–156, 1996.

[8] Y. Freund and R. E. Schapire. A decision-theoretic generalization of on-line learning and an application to boosting. *Journal of Comp. and Sys. Science*, 1997.

[9] J. Friedman, T. Hastie, and R. Tibshirani. Additive logistic regression: a statistical view of boosting. *Annals of Statistics*, 28, 1998.

[10] B. A. Frigyik, S. Srivastava, and M. R. Gupta. An introduction to functional derivatives. *Technical Report(University of Washington)*, 2008.

[11] Y. Guermeur. Vc theory of large margin multi-category classifiers. *J. Mach. Learn. Res.*, 8:2551–2594, December 2007.

[12] S. R. Ji Zhu, Hui Zou and T. Hastie. Multi-class adaboost. *Statistics and Its Interface*, 2:349–3660, 2009.

[13] H. Masnadi-Shirazi, N. Vasconcelos, and V. Mahadevan. On the design of robust classifiers for computer vision. In *CVPR*, 2010.

[14] L. Mason, J. Baxter, P. Bartlett, and M. Frean. Boosting algorithms as gradient descent. In *NIPS*, 2000.

[15] D. Mease and A. Wyner. Evidence contrary to the statistical view of boosting. *J. Mach. Learn. Res.*, 9:131–156, June 2008.

[16] I. Mukherjee and R. E. Schapire. A theory of multiclass boosting. In *NIPS*, 2010.

[17] M. J. Saberian, H. Masnadi-Shirazi, and N. Vasconcelos. Taylorboost: First and second order boosting algorithms with explicit margin control. In *CVPR*, 2010.

[18] R. E. Schapire and Y. Singer. Improved boosting algorithms using confidence-rated predictions. *Mach. Learn.*, 37:297–336, December 1999.

[19] V. N. Vapnik. *Statistical Learning Theory*. John Wiley Sons Inc, 1998.

